# Improving Transfer Rates in Brain Computer Interfacing: A Case Study

**Peter Meinicke, Matthias Kaper, Florian Hoppe, Manfred Heumann and Helge Ritter**
University of Bielefeld
Bielefeld, Germany
*{pmeinick, mkaper, fhoppe, helge} @techfak.uni-bielefeld.de*

## Abstract

In this paper we present results of a study on brain computer interfacing. We adopted an approach of Farwell & Donchin [4], which we tried to improve in several aspects. The main objective was to improve the transfer rates based on offline analysis of EEG-data but within a more realistic setup closer to an online realization than in the original studies. The objective was achieved along two different tracks: on the one hand we used state-of-the-art machine learning techniques for signal classification and on the other hand we augmented the data space by using more electrodes for the interface. For the classification task we utilized SVMs and, as motivated by recent findings on the learning of discriminative densities, we accumulated the values of the classification function in order to combine several classifications, which finally lead to significantly improved rates as compared with techniques applied in the original work. In combination with the data space augmentation, we achieved competitive transfer rates at an average of 50.5 bits/min and with a maximum of 84.7 bits/min.

## 1 Introduction

Some neurological diseases result in the so-called *locked-in syndrome*. People suffering from this syndrom lost control over their muscles, and therefore are unable to communicate. Consequently, their brain-signals should be used for communication. Besides the clinical application, developing such a *brain-computer interface* (BCI) is in itself an exciting goal as indicated by a growing research interest in this field.

Several EEG-based techniques have been proposed for realization of BCIs (see [6, 12], for an overview). There are at least four distinguishable basic approaches, each with its own advantages and shortcomings:

1. In the first approach, participants are trained to control their EEG frequency pattern for binary decisions. Whether specific frequencies (the $\mu$ and $\beta$ rhythms) in the power range are heightened or not results in upward or downward cursor movements. A further version extended this basic approach for 2D-movements. Transfer rates of 20-25 bits/min were reported [12].

2. Imaginations of movements, resulting in the "Bereitschaftspotential" over sensorimotor cortex areas, are used to transmit information in the device of Pfurtscheller

Figure 1: Stimulusmatrix with one column highlighted.

   et al. [8], which is in use by a tetraplegic patient. Blankertz et al. [2] applied
   sophisticated methods for data-analysis to this approach and reached fast transfer
   rates of 23 bits/min when classifying brain signals preceding overt muscle activity.

3. The thought translation device by Birbaumer et al. [5, 1] is based on slow cortical
   potentials, i.e. large shifts in the EEG-signal. They trained people in a biofeedback
   scenario to control this component. It is rather slow (<6 bits/min) and requires
   intensively trained participants but is in practical use.

4. Farwell & Donchin [4, 3, 10] developed a BCI-System by utilizing specific posi-
   tive deflections (P300) in EEG-signals accompanying rare events (as discussed in
   detail below). It is moderately fast (up to 12 bits/min) and needs no practice of the
   participant, but requires visual attention.

For BCIs, it is very desirable to have fast transfer rates. In our own studies, we therefore
tried to accelerate the fourth approach by using state-of-the-art machine learning techniques
and fusing data from different electrodes for data-analysis. For that purpose we utilized the
basic setup of Farwell & Donchin (referred to as F&D) [4] who used the well-studied
P300-Component to create a BCI-system. They presented a $6 \times 6$-matrix (see Fig. 1), filled
with letters and digits, and highlighted all rows and columns sequentially in random or-
der. People were instructed to focus on one symbol in the matrix, and mentally count its
highlightings. From EEG-research it is known, that counting a rare specific event (*oddball-
stimulus*) in a series of *background stimuli* evokes a P300 for the oddball stimulus. Hence,
highlighting the attended symbol in the $6 \times 6$-matrix should result in a P300, a character-
istic positive deflection with a latency of around 300ms in the EEG-signal. It is therefore
possible to infer the selected symbol by detecting the P300 in EEG-signals. Under suitable
circumstances, most brains expose a P300. Thus, no training of the participants is nec-
essary. For identification of the right column and row associated with a P300, Farwell &
Donchin used the model-based techniques *Area* and *Peak picking* (both described in section
2) to detect the P300. In addition, as a data-driven approach, they used *Stepwise Discrimi-
nant Analysis* (SWDA). Using SWDA in a later study [3] resulted in transfer rates between
4.8 and 7.8 symbols per minute at an accuracy of 80% with a temporal distance of 125ms
between two highlightings.

In our work reported here we could improve several aspects of the F&D-approach by utiliz-
ing very recent machine learning techniques and a larger number of EEG-electrodes. First
of all, we could increase the transfer rate by using *Support Vector Machines* (SVM) [11] for
classification. Inspired by a recent approach to learning of discriminative densities [7] we
utilized the values of the SVM classification function as a measure of confidence which we
accumulate over certain classifications in order to speed up the transfer rate. In addition,
we enhanced classification rates by augmenting the data-space. While Farwell & Donchin
employed only data from a single electrode for classification, we used the data from 10
electrodes simultaneously.

## 2 Methods

In the following we describe the techniques used for acquisition, preprocessing and analysis of the EEG-data.

**Data acquisition.** All results of this paper stem from offline analyses of data acquired during EEG-experiments. The experimental setup was the following: participants were seated in front of a computer screen presenting the matrix (see Fig. 1) and user instructions. EEG-data were recorded with 10 Ag/AgCl electrodes at positions of the extended international 10-20 system (Fz, Cz, Pz, C3, C4, P3, P4, Oz, OL, OR[1]) sampled at 200Hz and low-pass filtered at 30Hz. The participants had to perform a certain number of trials. For the duration of a trial, they were instructed

- to focus their attention on a target symbol specified by the program,
- to mentally count the highlightings of the target symbol, and
- to avoid any body movement (especially eye moves and blinks).

Each trial is subdivided into a certain number of subtrials. During each subtrial, 12 stimuli are presented, i.e. the 6 rows and the 6 columns are highlighted in random order. For different BCI-setups, the time between stimulus onsets, the interstimulus interval (ISI), was either 150, 300 or 500ms, while a highlighting always lasts 150ms. To each stimulus correspondes an *epoch*, a time frame of 600ms after stimulus onset [2] During this interval a P300 should be evoked if the stimulus contains the target symbol.

There is no pause between subtrials, but between trials. During the pause, the participants had time to focus on the next target symbol, before they initiated the next trial. The target symbol was chosen randomly from the available set of symbols and was presented by the program in order to create a data set of labelled EEG-signals for the subsequent offline analysis.

**Data preprocessing.** To compensate for slow drifts of the DC potential, in a first step the linear trend of the raw data in each electrode over the duration of a trial was eliminated. In a second step, the data was normalized to zero mean and unit standard deviation. This was separately done for each electrode taking the data of all trials into account.

**Classification of Epochs.** Test- and trainingsets were created by choosing the data according to one symbol as testset, and the data of the other symbols as trainingset in a crossvalidation scheme.

The task of classifying a subtrial for the identification of a target symbol has to be distinguished from the classification of a single epoch for detection of a signal, correlated with oddball-stimuli, which we briefly refer to as a "P300 component" in a simplified manner in the following. In case of using a subtrial to select a symbol, two P300 components have to be detected within epochs: one corresponding to a row-, another to a column-stimulus. The detection algorithm works on the data of an epoch and has to compute a *score* which reflects the presence of a P300 within that epoch. Therefore, 12 epochs have to be evaluated for the selection of one target symbol.

For the P300-detection, we utilized two model-based methods which had been proposed by F&D, and one completely data-driven method based on Support Vector Machines (SVMs) [11]. For training of the classifiers, we built up a sets of epochs containing an equal number of positive and negative examples, i.e. epochs with and without a P300 component.

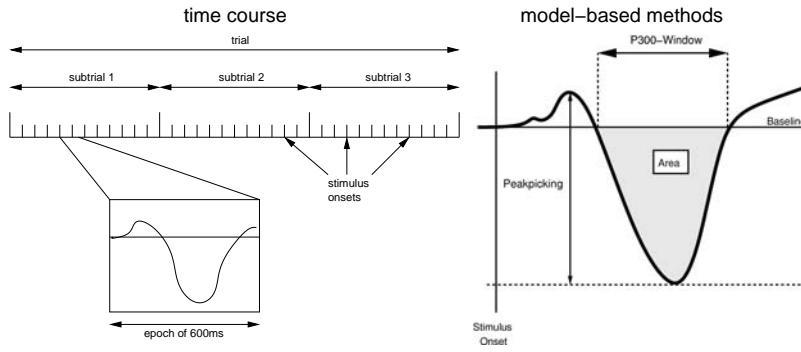

Figure 2: Trials, subtrials and epochs in the course of time (left). Model-based methods for analysis. Area calculates surface in the P300-window, Peak picking calculates differences between peaks.

The first model-based method uses as its score as shown in Fig. 2 the area in the P300-window ("Area method", $S_A$), while the second model-based method uses the difference between the lowest point before, and the highest point within the P300-window ("Peak picking method", $S_P$). Hyperparameters of the model-based methods were the boundaries of the P300-window. They were selected regarding the average of epochs containing the P300 by taking the boundaries of the largest area.

For the completely data-driven approach, SVMs were optimized to distinguish between the two classes (w/o P300) implied by the training set. As compared with many traditional classifiers, such as the SWDA method used by F&D, SVMs can realize Bayes-consistent classifiers under very general conditions without requiring any specific assumptions about the underlying data distributions and decision boundaries. Thereby convergence to the Bayes optimum can be achieved by a suitable choice of hyperparameters.

When using SVMs, it is not clear what measure to take as the score of an epoch. The problem is that the SVM has first of all been designed to assign binary class labels to its input without any measure of confidence on the resulting decision.

However, a recent approach to learning of discriminative densities [7] suggests an interpretation of the usual discrimination function for SVMs with positive kernels in terms of scaled density differences. This finding provides us with a well-motivated score of an epoch: with $\mathbf{x}$ as the data vector of an epoch and $y \in \{-1, 1\}$ as the corresponding class label which is positive/negative for epochs with/without target stimulus the SVM-score is computed as

$$S_{\text{SVM}} = f(\mathbf{x}) = \sum_i y_i \alpha_i K(\mathbf{x}, \mathbf{x}_i) + b \tag{1}$$

where $K(\cdot, \mathbf{x}_i)$ in our case is a Gaussian Kernel function with bandwidth $\sigma$ (selected as the weight $C$ for the soft-margin penalties by 3-fold crossvalidation) evaluated at the $i$-th data example. The mixing weights $\alpha_i$ were estimated by quadratic optimization for an SVM objective with linear soft-margin penalties where we used the SMO-algorithm [9].

**Combination of subtrials.** Because EEG-data possess a very poor signal-to-noise ratio (SNR), identification of the target symbol from a single subtrial is usually not reliable enough to achieve a reasonable classification rate. Therefore, several subtrials have to be combined for classification, slowing down the transfer rate. Thus, an important goal is to decrease the amount of subtrials which have to be combined for a satisfactory classification rate.

An important constraint for the development of the specific offline-analysis programs was to realize a testing scheme which should be as close as possible to a corresponding online evaluation. Therefore, we tested a method for certain $n$-combinations of subtrials in the following way: different series of $n$ successive subtrials were taken out of a test set and the corresponding single classifications were combined as explained below. Thereby, the test series contained only subtrials belonging to identical symbols and these were combined in their original temporal order[3].

In contrast, Farwell & Donchin randomly chose samples from a test set, built from subtrials taken from different trials and belonging to different symbols. With this procedure, they broke up the time course of the recorded data and did not distinguish between different symbols, i.e. different positions in the matrix on the screen.

Based on the data of $n$ subtrials, one has to choose a row and a column in order to identify the target symbol, i.e. to classify a trial. Therefore, in a first step, the single scores[4] $S(\mathbf{x}_{ik}^{(r)})$ of the epoch $\mathbf{x}_{ik}^{(r)}$ corresponding to the stimulus associated to the $i$-th row of the $k$-th subtrial were summed up to the total score $s_i^{(r)} = \sum_{k=1}^{n} S(\mathbf{x}_{ik}^{(r)})$. Then, the target row was chosen as $\arg\max_i s_i^{(r)}$ with $i = 1, \ldots, 6$. Equivalent steps were performed to choose the target column. Based on these decisions the target symbol was finally selected in accordance to the presented matrix.

## 3  Experimental Results

Before going into details, we outline our investigations about improving the usability of the F&D-BCI. First, the different methods were compared to classify the data of the Pz electrode, which was originally used by Farwell & Donchin. Second, further single electrodes were taken as input source. This revealed information about interesting scalp positions to record a P300 and on the other hand indicated which channels may contain a useful signal. Third, the SVM classification rate with respect to epochs was improved by increasing the data-space. Therefore, the input vector for the classifier was extended by combining data from the same epoch but from different electrodes. These tests indicated that the best classification rates could be achieved using as detection method an SVM with all ten electrodes as input sources.

Since the results of the first three steps were established based on the data of one initial experiment with only one participant, we evaluated the generality of these techniques by testing different subjects and BCI parameters. Finally, the BCI performance in terms of attainable communication rates is estimated from these analyses.

**Method comparison using the Pz electrode as input source.**  All four methods were applied to the data of one initial experiment with an ISI of 500ms and 3 subtrials per trial. Figure 3 presents the classification rates of up to 10 subtrials.

The SVM method achieved best performance, its epoch classification rate was 76.3% (SD=1.0) in a 10-fold crossvalidation with about 380 subtrials samples in the training sets, and about 40 in the test sets. Of each subtrial in the training set, 4 epochs (2 with, 2 without a P300) were taken as training samples, whereas all 12 epochs of the subtrials of the test set were classified. For each training set, hyperparameters were selected by another 3-fold crossvalidation on this set.

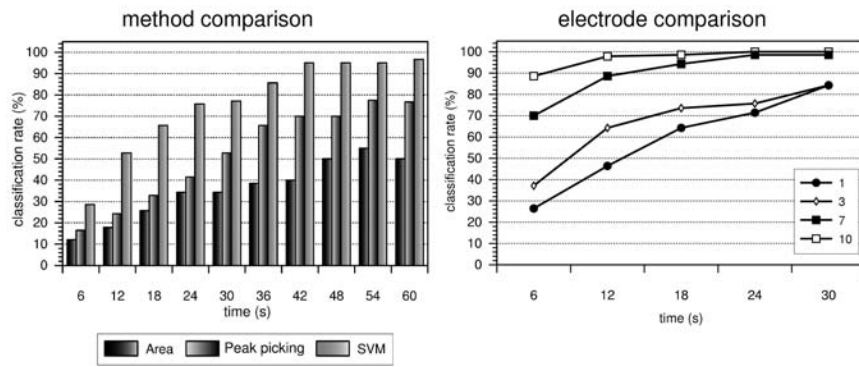

Figure 3: (left) Method comparison on the Pz electrode: The three techniques were applied to the data of the initial experiment. (right) Classification rates for different number of electrodes.

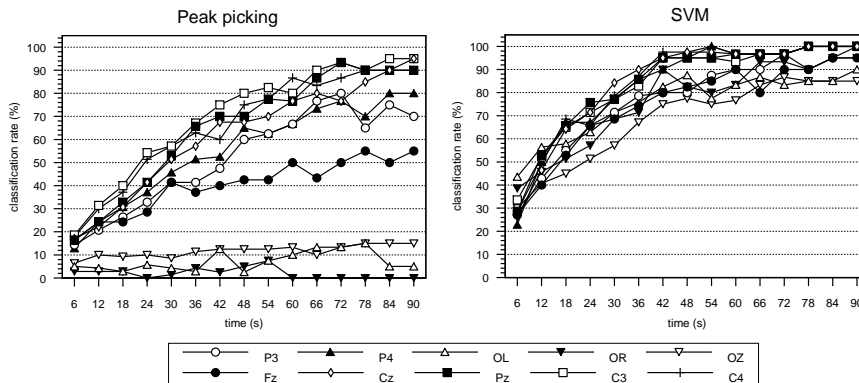

Figure 4: Electrode comparison on the data of the initial experiment.

**Different electrodes as input source.** The method comparison tests were repeated for each electrode. The results of the Peak picking and SVM method are shown in Figure 3.

The SVM is able to extract useful information from all ten electrodes, whereas the Peak picking performance varies for different scalp positions. Especially, the electrodes over the visual cortex areas OZ, OR and OL are useless for the model-based techniques, as the same characteristics are revealed by tests with the Area method.

**Higher-dimensional data-space.** While Farwell & Donchin used only one electrode for data-analysis, we extended the data-space by using larger numbers of electrodes. We calculated classification rates for Pz alone, three, seven, and ten electrodes. A signal correlated with oddball-stimuli was classified at rates of 76.8%, 76.8%, 90.9%, and 94.5%, respectively for the different data-spaces of 120, 360, 840, and 1200 dimensions. These rates were calculated with 850 positive and 850 negative epoch samples and a 3-fold crossvalidation. This classified signal might be more than solely the traditional P300 component. Applying data-space augmentation for classification to infer symbols in the matrix results in the classification rates depicted in Figure 3 (right) for an ISI of 500ms. Using ten electrodes simultaneously, combined in one data vector, outperforms lower-dimensional data-spaces.

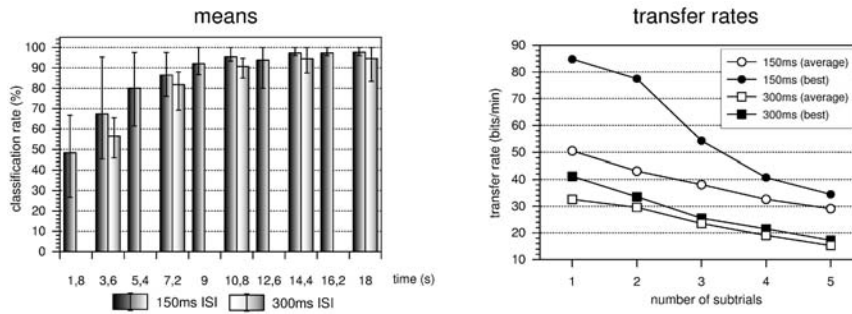

Figure 5: Mean-classification rates (left) and transfer rates (right) for different ISIs. Error bars range from best to worst results. Note that a subtrial takes a specific amount of time. Therefore, the time dependend transfer rates are decreasing with the number of subtrials.

**Reducing the ISI and using more participants.** The improved classification rates encouraged further experiments. To accelerate the system, we reduced the ISI to 300ms and 150ms. Additionally, to generalize the results, we recruited four participants. Means, best and worst classification rates are presented in Figure 5, as well as average and best transfer rates. The latter were calculated according to

$$B_t = t/60(\log_2 N + p \log_2 p + (1 - p) \log_2(1 - p/N - 1)),$$

where $N$ is the number of choices (36 here), $p$ the probability for classification, and $t$ the time required for classification.

Using an ISI of 300ms results in slower transfer rates than using an ISI of 150ms. The latter ISI results on the average in classifying a symbol after 5.4s with an accuracy of 80% (disregarding delays between trials). The poorest performer needs 9s to reach this criterion, the best performer achieves an accuracy of 95.2% already after 3.6s. The transfer rates, with a maximum of 84.7 bits/min and an average of 50.5 bits/min outperform the EEG-based BCI-systems we know.

## 4   Conclusion

With an application of the data-driven SVM-method to classification of single-channel EEG-signals, we could improve transfer rates as compared with model-based techniques. Furthermore, by increasing the number of EEG-channels, even higher classification and transfer rates could be achieved. Accumulating the value of the classification function as measure of confidence proved to be practical to handle series of classifications in order to identify a symbol. This resulted in high transfer rates with a maximum of 84.7 bits/min.

## 5   Acknowledgements

We thank Thorsten Twellmann for supplying the SVM-algorithms and the Department of Cognitive Psychology at the University of Bielefeld for providing the experimental environment. This work was supported by Grant Ne 366/4-1 and the project SFB 360 from the German Research Council (Deutsche Forschungsgemeinschaft).

## Footnotes

[1]OL denotes the position halfway between O1 and T5, and OR between O2 and T6 respectively.

[2]With an ISI shorter than 450ms, there is a time overlap of consecutive epochs.

[3]For a higher number of subtrial combinations, subtrials from different trials had to be combined. However, real-world-application of this BCI don't require such combinations with respect to the finally achieved transfer rates reported in section 3.

[4]The method index is omitted in the following.

# References

[1] N. Birbaumer, N. Ghanayim, T. Hinterberger, I. Iversen, B. Kotchoubey, A. Kübler, J. Perelmouter, E. Taub, and H. Flor. A spelling device for the paralysed. *Nature*, 398:297–298, 1999.

[2] B. Blankertz, G. Curio, and K.-R. Müller. Classifying single trial eeg: Towards brain computer interfacing. In T. G. Dietterich, S. Becker, and Z. Ghahramani, editors, *Advances in Neural Information Processing Systems 14*, Cambridge, MA, 2002. MIT Press.

[3] E. Donchin, K.M. Spencer, and R. Wijeshinghe. The mental prosthesis: Assessing the speed of a p300-based brain-computer interface. *IEEE Transactions on Rehabilitation Engineering*, 8(2):174–179, 2000.

[4] L.A. Farwell and E. Donchin. Talking off the top of your head: toward a mental prosthesis utilizing event-related brain potentials. *Electroencephalography and clinical Neurophysiology*, 70(S2):510–523, 1988.

[5] A. Kübler, B. Kotchoubey, T. Hinterberger, N. Ghanayim, J. Perelmouter, M. Schauer, C. Fritsch, E. Taub, and N. Birbaumer. The thought translation device: a neurophysiological approach to commincation in total motor paralysis. *Experimental Brain Research*, 124:223–232, 1999.

[6] A. Kübler, B. Kotchoubey, J. Kaiser, J.R. Wolpaw, and N. Birbaumer. Brain-computer communication: Unlocking the locked in. *Psychological Bulletin*, 127(3):358–375, 2001.

[7] P. Meinicke, T. Twellmann, and H. Ritter. Maximum contrast classifiers. In *Proc. of the Int. Conf. on Artificial Neural Networks*, Berlin, 2002. Springer. in press.

[8] G. Pfurtscheller, C. Neuper, C. Guger, B. Obermaier, M. Pregenzer, H. Ramoser, and A. Schlögl. Current trends in graz brain-computer interface (bci) research. *IEEE Transactions On Rehabilitation Engineering*, pages 216–219, 2000.

[9] J. Platt. Fast training of support vector machines using sequential minimal optimization. In B. Schölkopf, C. J. C. Burges, and A. J. Smola, editors, *Advances in Kernel Methods — Support Vector Learning*, pages 185–208, Cambridge, MA, 1999. MIT Press.

[10] J.B. Polikoff, H.T. Bunnell, and W.J. Borkowski. Toward a p300-based computer interface. *RESNA '95 Annual Conference and RESNAPRESS and Arlington Va.*, pages 178–180, 1995.

[11] V. N. Vapnik. *The Nature of Statistical Learning Theory*. Springer, New York, 1995.

[12] J.R. Wolpaw, N. Birbaumer, D.J. McFarland, G. Pfurtscheller, and T.M. Vaughan. Brain-computer interfaces for communication and control. *Clinical Neurophysiology*, 113:767–791, 2002.
